# A classification-based cocktail-party processor

**Nicoleta Roman, DeLiang Wang**
Department of Computer and Information
Science and Center for Cognitive Science
The Ohio State University
Columbus, OH  43210, USA
{niki,dwang}@cis.ohio-state.edu

**Guy J. Brown**
Department of Computer Science
University of Sheffield
211 Portobello Street
Sheffield, S1 4DP, UK
g.brown@dcs.shef.ac.uk

## Abstract

At a cocktail party, a listener can selectively attend to a single voice and filter out other acoustical interferences. How to simulate this perceptual ability remains a great challenge. This paper describes a novel supervised learning approach to speech segregation, in which a target speech signal is separated from interfering sounds using spatial location cues: interaural time differences (ITD) and interaural intensity differences (IID). Motivated by the auditory masking effect, we employ the notion of an ideal time-frequency binary mask, which selects the target if it is stronger than the interference in a local time-frequency unit. Within a narrow frequency band, modifications to the relative strength of the target source with respect to the interference trigger systematic changes for estimated ITD and IID. For a given spatial configuration, this interaction produces characteristic clustering in the binaural feature space. Consequently, we perform pattern classification in order to estimate ideal binary masks. A systematic evaluation in terms of signal-to-noise ratio as well as automatic speech recognition performance shows that the resulting system produces masks very close to ideal binary ones. A quantitative comparison shows that our model yields significant improvement in performance over an existing approach. Furthermore, under certain conditions the model produces large speech intelligibility improvements with normal listeners.

## 1  Introduction

The perceptual ability to detect, discriminate and recognize one utterance in a background of acoustic interference has been studied extensively under both monaural and binaural conditions [1, 2, 3]. The human auditory system is able to segregate a speech signal from an acoustic mixture using various cues, including fundamental frequency (F0), onset time and location, in a process that is known as

auditory scene analysis (ASA) [1]. F0 is widely used in computational ASA systems that operate upon monaural input – however, systems that employ only this cue are limited to voiced speech [4, 5, 6]. Increased speech intelligibility in binaural listening compared to the monaural case has prompted research in designing cocktail-party processors based on spatial cues [7, 8, 9]. Such a system can be applied to, among other things, enhancing speech recognition in noisy environments and improving binaural hearing aid design.

In this study, we propose a sound segregation model using binaural cues extracted from the responses of a KEMAR dummy head that realistically simulates the filtering process of the head, torso and external ear. A typical approach for signal reconstruction uses a time-frequency (T-F) mask: T-F units are weighted selectively in order to enhance the target signal. Here, we employ an ideal binary mask [6], which selects the T-F units where the signal energy is greater than the noise energy. The ideal mask notion is motivated by the human auditory masking phenomenon, in which a stronger signal masks a weaker one in the same critical band. In addition, from a theoretical ASA perspective, an ideal binary mask gives a performance ceiling for all binary masks. Moreover, such masks have been recently shown to provide a highly effective front-end for robust speech recognition [10]. We show for mixtures of multiple sound sources that there exists a strong correlation between the relative strength of target and interference and estimated ITD/IID, resulting in a characteristic clustering across frequency bands. Consequently, we employ a nonparametric classification method to determine decision regions in the joint ITD-IID feature space that correspond to an optimal estimate for an ideal mask.

Related models for estimating target masks through clustering have been proposed previously [11, 12]. Notably, the experimental results by Jourjine et al. [12] suggest that speech signals in a multiple-speaker condition obey to a large extent disjoint orthogonality in time and frequency. That is, at most one source has a nonzero energy at a specific time and frequency. Such models, however, assume input directly from microphone recordings and head-related filtering is not considered. Simulation of human binaural hearing introduces different constraints as well as clues to the problem. First, both ITD and IID should be utilized since IID is more reliable at higher frequencies than ITD. Second, frequency-dependent combinations of ITD and IID arise naturally for a fixed spatial configuration. Consequently, channel-dependent training should be performed for each frequency band.

The rest of the paper is organized as follows. The next section contains the architecture of the model and describes our method for azimuth localization. Section 3 is devoted to ideal binary mask estimation, which constitutes the core of the model. Section 4 presents the performance of the system and a quantitative comparison with the Bodden [7] model. Section 5 concludes our paper.

## 2   Model architecture and azimuth localization

Our model consists of the following stages: 1) a model of the auditory periphery; 2) frequency-dependent ITD/IID extraction and azimuth localization; 3) estimation of an ideal binary mask.

The input to our model is a mixture of two or more signals presented at different, but fixed, locations. Signals are sampled at 44.1 kHz. We follow a standard procedure for simulating free-field acoustic signals from monaural signals (no reverberations are modeled). Binaural signals are obtained by filtering the monaural signals with measured head-related transfer functions (HRTF) from a KEMAR dummy head [13]. HRTFs introduce a natural combination of ITD and IID into the signals that is extracted in the subsequent stages of the model.

To simulate the auditory periphery we use a bank of 128 gammatone filters in the range of 80 Hz to 5 kHz as described in [4]. In addition, the gains of the gammatone filters are adjusted in order to simulate the middle ear transfer function. In the final step of the peripheral model, the output of each gammatone filter is half-wave rectified in order to simulate firing rates of the auditory nerve. Saturation effects are modeled by taking the square root of the signal.

Current models of azimuth localization almost invariably start with Jeffress's cross-correlation mechanism. For all frequency channels, we use the normalized cross-correlation computed at lags equally distributed in the plausible range from –1 ms to 1 ms using an integration window of 20 ms. Frequency-dependent nonlinear transformations are used to map the time-delay axis onto the azimuth axis resulting in a cross-correlogram structure. In addition, a 'skeleton' cross-correlogram is formed by replacing the peaks in the cross-correlogram with Gaussians of narrower widths that are inversely proportional to the channel center frequency. This results in a sharpening effect, similar in principle to lateral inhibition. Assuming fixed sources, multiple locations are determined as peaks after summating the skeleton cross-correlogram across frequency and time. The number of sources and their locations computed here, as well as the target source location, feed to the next stage.

## 3   Binary mask estimation

The objective of this stage of the model is to develop an efficient mechanism for estimating an ideal binary mask based on observed patterns of extracted ITD and IID features. Our theoretical analysis for two-source interactions in the case of pure tones shows relatively smooth changes for ITD and IID with the relative strength $R$ between the two sources in narrow frequency bands [14]. More specifically, when the frequencies vary uniformly in a narrow band the derived mean values of ITD/IID estimates vary monotonically with respect to $R$.

To capture this relationship in the context of real signals, statistics are collected for individual spatial configurations during training. We employ a training corpus consisting of 10 speech utterances from the TIMIT database (see [14] for details). In the two-source case, we divide the corpus in two equal sets: target and interference. In the three-source case, we select 4 signals for the target set and 2 interfering sets of 3 signals each.

For all frequency channels, local estimates of ITD, IID and $R$ are based on 20-ms time frames with 10 ms overlap between consecutive time frames. In order to eliminate the multi-peak ambiguity in the cross-correlation function for mid- and high-frequency channels, we use the following strategy. We compute $\text{ITD}_i$ as the peak location of the cross-correlation in the range $2\pi / \omega_i$ centered at the target ITD, where $\omega_i$ indicates the center frequency of the $i$th channel. On the other hand, IID and $R$ are computed as follows:

$$\text{IID}_i = 20 \log_{10} \sum_t r_i^2(t) \bigg/ \sum_t l_i^2(t), \quad R_i = \sqrt{\sum_t s_i^2(t)} \bigg/ \left( \sqrt{\sum_t s_i^2(t)} + \sqrt{\sum_t n_i^2(t)} \right)$$

where $l_i$ and $r_i$ refer to the left and right peripheral output of the $i$th channel, respectively, $s_i$ refers to the output for the target signal, and $n_i$ that for the acoustic interference. In computing $\text{IID}_i$, we use 20 instead of 10 in order to compensate for the square root operation in the peripheral model.

Fig. 1 shows empirical results obtained for a two-source configuration on the training corpus. The data exhibits a systematic shift for both ITD and IID with respect to the relative strength $R$. Moreover, the theoretical mean values obtained in the case of pure tones [14] match the empirical ones very well. This observation extends to multiple-source scenarios. As an example, Fig. 2 displays histograms that show the relationship between $R$ and both ITD (Fig. 2A) and IID (Fig. 2B) for a three-source situation. Note that the interfering sources introduce systematic deviations for the binaural cues. Consider a worst case: the target is silent and two interferences have equal energy in a given T-F unit. This results in binaural cues indicating an auditory event at half of the distance between the two interference locations; for Fig. 2, it is 0° - the target location. However, the data in Fig. 2 has a low probability for this case and shows instead a clustering phenomenon, suggesting that in most cases only one source dominates a T-F unit.

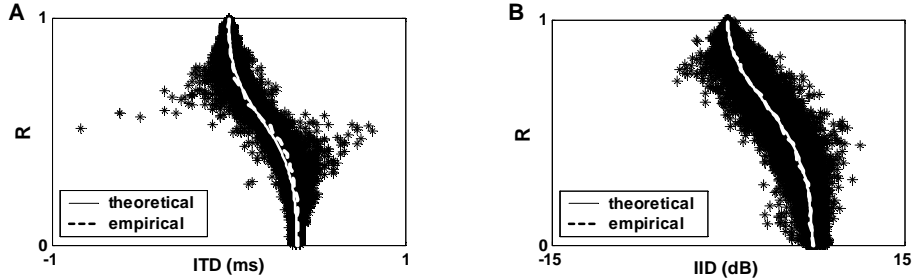

**Figure 1**. Relationship between ITD/IID and relative strength $R$ for a two-source configuration: target in the median plane and interference on the right side at 30°. The solid curve shows the theoretical mean and the dash curve shows the data mean. **A**: The scatter plot of ITD and $R$ estimates for a filter channel with center frequency 500 Hz. **B**: Results for IID for a filter channel with center frequency 2.5 kHz.

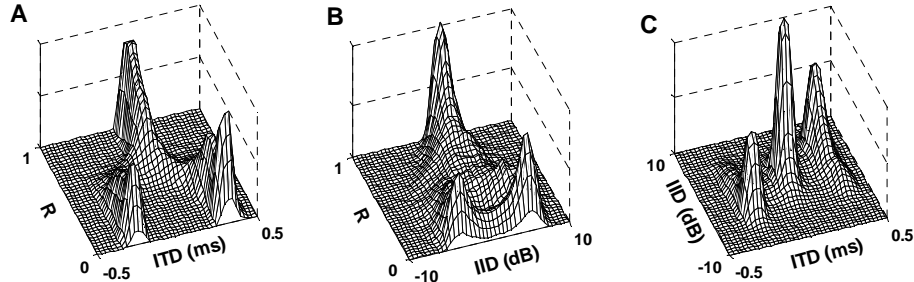

**Figure 2**. Relationship between ITD/IID and relative strength $R$ for a three-source configuration: target in the median plane and interference at -30° and 30°. Statistics are obtained for a channel with center frequency 1.5 kHz. **A**: Histogram of ITD and $R$ samples. **B**: Histogram of IID and $R$ samples. **C**: Clustering in the ITD-IID space.

By displaying the information in the joint ITD-IID space (Fig. 2C), we observe location-based clustering of the binaural cues, which is clearly marked by strong peaks that correspond to distinct active sources. There exists a tradeoff between ITD and IID across frequencies, where ITD is most salient at low frequencies and IID at high frequencies [2]. But a fixed cutoff frequency that separates the effective use of ITD and IID does not exist for different spatial configurations. This motivates our choice of a joint ITD-IID feature space that optimizes the system performance across different configurations. Differential training seems necessary for different channels given that there exist variations of ITD and, especially, IID values for different center frequencies.

Since the goal is to estimate an ideal binary mask, we focus on detecting decision regions in the 2-dimensional ITD-IID space for individual frequency channels.

Consequently, supervised learning techniques can be applied. For the $i$th channel, we test the following two hypotheses. The first one is $H_1$: target is dominant or $R_i > 0.5$, and the second one is $H_2$: interference is dominant or $R_i < 0.5$. Based on the estimates of the bivariate densities $p(x \mid H_1)$ and $p(x \mid H_2)$ the classification is done by the *maximum a posteriori* decision rule: $p(H_1)p(x \mid H_1) > p(H_2)p(x \mid H_2)$. There exist a plethora of techniques for probability density estimation ranging from parametric techniques (e.g. mixture of Gaussians) to nonparametric ones (e.g. kernel density estimators). In order to completely characterize the distribution of the data we use the kernel density estimation method independently for each frequency channel. One approach for finding smoothing parameters is the least-squares cross-validation method, which is utilized in our estimation.

One cue not employed in our model is the interaural time difference between signal envelopes (IED). Auditory models generally employ IED in the high-frequency range where the auditory system becomes gradually insensitive to ITD. We have compared the performance of the three binaural cues: ITD, IID and IED and have found no benefit for using IED in our system after incorporating ITD and IID [14].

## 4   Performance and comparison

The performance of a segregation system can be assessed in different ways, depending on intended applications. To extensively evaluate our model, we use the following three criteria: 1) a signal-to-noise (SNR) measure using the original target as signal; 2) ASR rates using our model as a front-end; and 3) human speech intelligibility tests.

To conduct the SNR evaluation a segregated signal is reconstructed from a binary mask using a resynthesis method described in [5]. To quantitatively assess system performance, we measure the SNR using the original target speech as signal:

$$SNR = 10 \log_{10} \sum_t s_o^2(t) \bigg/ \sum_t \left(s_o(t) - s_e(t)\right)^2$$

where $s_o(t)$ represents the resynthesized original speech and $s_e(t)$ the reconstructed speech from an estimated mask. One can measure the initial SNR by replacing the denominator with $s_N(t)$, the resynthesized original interference.

Fig. 3 shows the systematic results for two-source scenarios using the Cooke corpus [4], which is commonly used in sound separation studies. The corpus has 100 mixtures obtained from 10 speech utterances mixed with 10 types of intrusion. We compare the SNR gain obtained by our model against that obtained using the ideal binary mask across different noise types. Excellent results are obtained when the target is close to the median plane for an azimuth separation as small as 5°. Performance degrades when the target source is moved to the side of the head, from an average gain of 13.7 dB for the target in the median plane (Fig. 3A) to 1.7 dB when target is at 80° (Fig. 3B). When spatial separation increases the performance improves even for side targets, to an average gain of 14.5 dB in Fig. 3C. This performance profile is in qualitative agreement with experimental data [2].

Fig. 4 illustrates the performance in a three-source scenario with target in the median plane and two interfering sources at –30° and 30°. Here 5 speech signals from the Cooke corpus form the target set and the other 5 form one interference set. The second interference set contains the 10 intrusions. The performance degrades compared to the two-source situation, from an average SNR of about 12 dB to 4.1

dB. However, the average SNR gain obtained is approximately 11.3 dB. This ability of our model to segregate mixtures of more than two sources differs from blind source separation with independent component analysis.

In order to draw a quantitative comparison, we have implemented Bodden's cocktail-party processor using the same 128-channel gammatone filterbank [7]. The localization stage of this model uses an extended cross-correlation mechanism based on contralateral inhibition and it adapts to HRTFs. The separation stage of the model is based on estimation of the weights for a Wiener filter as the ratio between a desired excitation and an actual one. Although the Bodden model is more flexible by incorporating aspects of the precedence effect into the localization stage, the estimation of Wiener filter weights is less robust than our binary estimation of ideal masks. Shown in Fig. 5, our model shows a considerable improvement over the Bodden system, producing a 3.5 dB average improvement.

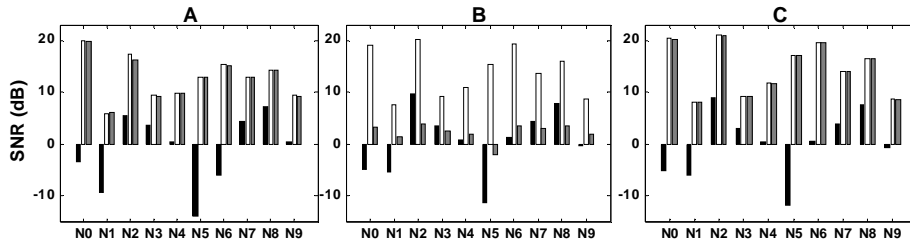

**Figure 3**. Systematic results for two-source configuration. Black bars correspond to the SNR of the initial mixture, white bars indicate the SNR obtained using ideal binary mask, and gray bars show the SNR from our model. Results are obtained for speech mixed with ten intrusion types (N0: pure tone; N1: white noise; N2: noise burst; N3: 'cocktail party'; N4: rock music; N5: siren; N6: trill telephone; N7: female speech; N8: male speech; N9: female speech). **A**: Target at 0°, interference at 5°. **B**: Target at 80°, interference at 85°. **C**: Target at 60°, interference at 90°.

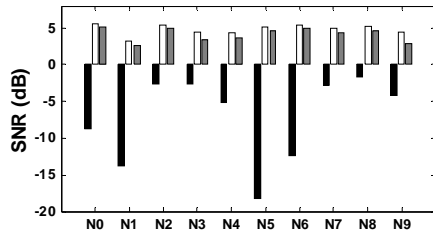 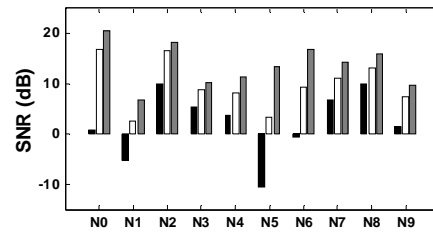

**Figure 4**. Evaluation for a three-source configuration: target at 0° and two interfering sources at –30° and 30°. Black bars correspond to the SNR of the initial mixture, white bars to the SNR obtained using the ideal binary mask, and gray bars to the SNR from our model.

**Figure 5**. SNR comparison between the Bodden model (white bars) and our model (gray bars) for a two-source configuration: target at 0° and interference at 30°. Black bars correspond to the SNR of the initial mixture.

For the ASR evaluation, we use the missing-data technique as described in [10]. In this approach, a continuous density hidden Markov model recognizer is modified such that only acoustic features indicated as reliable in a binary mask are used during decoding. Hence, it works seamlessly with the output from our speech segregation system. We have implemented the missing data algorithm with the same 128-channel gammatone filterbank. Feature vectors are obtained using the Hilbert envelope at the output of the gammatone filter. More specifically, each feature vector is extracted by smoothing the envelope using an 8-ms first-order filter, sampling at a frame-rate of 10 ms and finally log-compressing. We use the bounded marginalization method for classification [10]. The task domain is recognition of

connected digits, and both training and testing are performed on acoustic features from the left ear signal using the male speaker dataset in the TIDigits database.

Fig. 6A shows the correctness scores for a two-source condition, where the male target speaker is located at 0° and the interference is another male speaker at 30°. The performance of our model is systematically compared against the ideal masks for four SNR levels: 5 dB, 0 dB, -5 dB and –10 dB. Similarly, Fig. 6B shows the results for the three-source case with an added female speaker at -30°. The ideal mask exhibits only slight and gradual degradation in recognition performance with decreasing SNR and increasing number of sources. Observe that large improvements over baseline performance are obtained across all conditions. This shows the strong potential of applying our model to robust speech recognition.

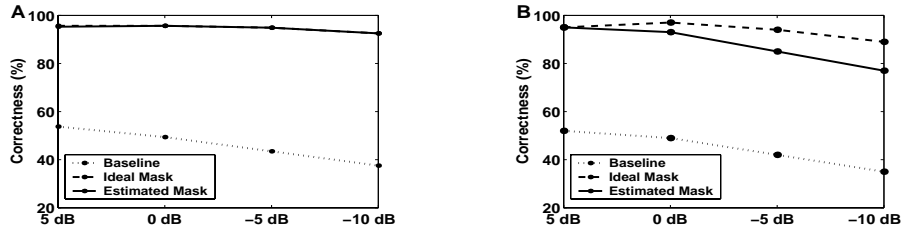

**Figure 6**. Recognition performance at different SNR values for original mixture (dotted line), ideal binary mask (dashed line) and estimated mask (solid line). **A**. Correctness score for a two-source case. **B**. Correctness score for a three-source case.

Finally we evaluate our model on speech intelligibility with listeners with normal hearing. We use the Bamford-Kowal-Bench sentence database that contains short semantically predictable sentences [15]. The score is evaluated as the percentage of keywords correctly identified, ignoring minor errors such as tense and plurality. To eliminate potential location-based priming effects we randomly swap the locations for target and interference for different trials. In the unprocessed condition, binaural signals are produced by convolving original signals with the corresponding HRTFs and the signals are presented to a listener dichotically. In the processed condition, our algorithm is used to reconstruct the target signal at the better ear and results are presented diotically.

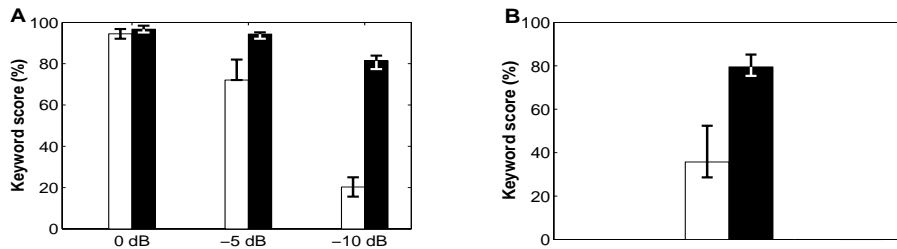

**Figure 7.** Keyword intelligibility score for twelve native English speakers (median values and interquartile ranges) before (white bars) and after processing (black bars). **A.** Two-source condition (0° and 5°). **B.** Three-source condition (0°, 30° and -30°).

Fig. 7A gives the keyword intelligibility score for a two-source configuration. Three SNR levels are tested: 0 dB, -5 dB and –10 dB, where the SNR is computed at the better ear. Here the target is a male speaker and the interference is babble noise. Our algorithm improves the intelligibility score for the tested conditions and the improvement becomes larger as the SNR decreases (61% at –10 dB). Our informal observations suggest, as expected, that the intelligibility score improves for unprocessed mixtures when two sources are more widely separated than 5°. Fig. 7B shows the results for a three-source configuration, where our model yields a 40%

improvement. Here the interfering sources are one female speaker and another male speaker, resulting in an initial SNR of –10 dB at the better ear.

## 5  Conclusion

We have observed systematic deviations of the ITD and IID cues with respect to the relative strength between target and acoustic interference, and configuration-specific clustering in the joint ITD-IID feature space. Consequently, supervised learning of binaural patterns is employed for individual frequency channels and different spatial configurations to estimate an ideal binary mask that cancels acoustic energy in T-F units where interference is stronger. Evaluation using both SNR and ASR measures shows that the system estimates ideal binary masks very well. A comparison shows a significant improvement in performance over the Bodden model. Moreover, our model produces substantial speech intelligibility improvements for two and three source conditions.

### Acknowledgments

This research was supported in part by an NSF grant (IIS-0081058) and an AFOSR grant (F49620-01-1-0027). A preliminary version of this work was presented in 2002 ICASSP.

### References

[1] A. S. Bregman, *Auditory Scene Analysis*, Cambridge, MA: MIT press, 1990.

[2] J. Blauert, *Spatial Hearing - The Psychophysics of Human Sound Localization,* Cambridge, MA: MIT press, 1997.

[3] A. Bronkhorst, "The cocktail party phenomenon: a review of research on speech intelligibility in multiple-talker conditions," *Acustica*, vol. 86, pp. 117-128, 2000.

[4] M. P. Cooke, *Modeling Auditory Processing and Organization,* Cambridge, U.K.: Cambridge University Press, 1993.

[5] G. J. Brown and M. P. Cooke, "Computational auditory scene analysis," *Computer Speech and Language*, vol. 8, pp. 297-336, 1994.

[6] G. Hu and D. L. Wang, "Monaural speech separation," *Proc. NIPS,* 2002.

[7] M. Bodden, "Modeling human sound-source localization and the cocktail-party-effect," *Acta Acoustica*, vol. 1, pp. 43-55, 1993.

[8] C. Liu *et al.*, "A two-microphone dual delay-line approach for extraction of a speech sound in the presence of multiple interferers," *J. Acoust. Soc. Am.,* vol. 110, pp. 3218-3230, 2001.

[9] T. Whittkop and V. Hohmann, "Strategy-selective noise reduction for binaural digital hearing aids," *Speech Comm.*, vol. 39, pp. 111-138, 2003.

[10] M. P. Cooke, P. Green, L. Josifovski and A. Vizinho, "Robust automatic speech recognition with missing and unreliable acoustic data," *Speech Comm.*, vol. 34, pp. 267-285, 2001.

[11] H. Glotin, F. Berthommier and E. Tessier, "A CASA-labelling model using the localisation cue for robust cocktail-party speech recognition," *Proc. EUROSPEECH*, pp. 2351-2354, 1999.

[12] A. Jourjine, S. Rickard and O. Yilmaz, "Blind separation of disjoint orthogonal signals: demixing N sources from 2 mixtures," *Proc. ICASSP*, 2000.

[13] W. G. Gardner and K. D. Martin, "HRTF measurements of a KEMAR dummy-head microphone," *MIT Media Lab Technical Report* #280, 1994.

[14] N. Roman, D. L. Wang and G. J. Brown, "Speech segregation based on sound localization," *J. Acoust. Soc. Am.,* vol. 114, pp. 2236-2252, 2003.

[15] J. Bench and J. Bamford, *Speech Hearing Tests and the Spoken Language of Hearing-Impaired Children*, London: Academic press, 1979.
